# Markov Networks for Detecting Overlapping Elements in Sequence Data

**Joseph Bockhorst**
Dept. of Computer Sciences
University of Wisconsin
Madison, WI 53706
*joebock@cs.wisc.edu*

**Mark Craven**
Dept. of Biostatistics and Medical Informatics
University of Wisconsin
Madison, WI 53706
*craven@biostat.wisc.edu*

## Abstract

Many sequential prediction tasks involve locating instances of patterns in sequences. Generative probabilistic language models, such as hidden Markov models (HMMs), have been successfully applied to many of these tasks. A limitation of these models however, is that they cannot naturally handle cases in which pattern instances overlap in arbitrary ways. We present an alternative approach, based on conditional Markov networks, that can naturally represent arbitrarily overlapping elements. We show how to efficiently train and perform inference with these models. Experimental results from a genomics domain show that our models are more accurate at locating instances of overlapping patterns than are baseline models based on HMMs.

## 1 Introduction

Hidden Markov models (HMMs) and related probabilistic sequence models have been among the most accurate methods used for sequence-based prediction tasks in genomics, natural language processing and other problem domains. One key limitation of these models, however, is that they cannot represent general overlaps among sequence elements in a concise and natural manner. We present a novel approach to modeling and predicting overlapping sequence elements that is based on undirected Markov networks. Our work is motivated by the task of predicting DNA sequence elements involved in the regulation of gene expression in bacteria. Like HMM-based methods, our approach is able to represent and exploit relationships among different sequence elements of interest. In contrast to HMMs, however, our approach can naturally represent sequence elements that overlap in arbitrary ways.

We describe and evaluate our approach in the context of predicting a bacterial genome's genes and regulatory "signals" (together its *regulatory elements*). Part of the process of understanding a given genome is to assemble a "parts list", often using computational methods, of its regulatory elements. Predictions, in this case, entail specifying the start and end coordinates of subsequences of interest. It is common in bacterial genomes for these important sequence elements to overlap.

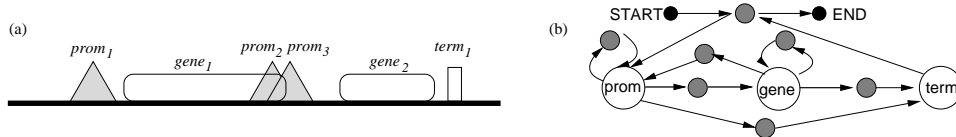

Figure 1: (a) Example arrangement of two genes, three promoters and one terminator in a DNA sequence. (b) Topology of an HMM for predicting these elements. Large circles represent element-specific sub-models and small gray circles represent inter-element sub-models, one for each allowed pair of adjacent elements. Due to the overlapping elements, there is no path through the HMM consistent with the configuration in (a).

Our approach to predicting overlapping sequence elements, which is based on discriminatively trained undirected graphical models called *conditional Markov networks* [5, 10] (also called *conditional random fields*), uses two key steps to make a set of predictions. In the first step, candidate elements are generated by having a set of models independently make predictions. In the second step, a Markov network is constructed to decide which candidate predictions to accept.

Consider the task of predicting *gene*, *promoter*, and *terminator* elements encoded in bacterial DNA. Figure 1(a) shows an example arrangement of these elements in a DNA sequence. Genes are DNA sequences that encode information for constructing proteins. Promoters and terminators are DNA sequences that regulate *transcription*, the first step in the synthesis of a protein from a gene. Transcription begins at a promoter, proceeds *downstream* (left-to-right in Figure 1(a)), and ends at a terminator. Regulatory elements often overlap each other, for example $prom_2$ and $prom_3$ or $gene_1$ and $prom_2$ in Figure 1.

One technique for predicting these elements is first to train a probabilistic sequence model for each element type (e.g. [2, 9]) and then to "scan" an input sequence with each model in turn. Although this approach can predict overlapping elements, it is limited since it ignores inter-element dependencies. Other methods, based on HMMs (e.g. [11, 1]), explicitly consider these dependencies. Figure 1(b) shows an example topology of such an HMM. Given an input sequence, this HMM defines a probability distribution over *parses*, partitionings of the sequence into subsequences corresponding to elements and the regions between them. These models are not naturally suited to representing overlapping elements. For the case shown in Figure 1(a) for example, even if the subsequences for $gene_1$ and $prom_2$ match their respective sub-models very well, since both elements cannot be in the same parse there is a competition between predictions of $gene_1$ and $prom_2$. One could expand the state set to include states for specific overlap situations however, the number of states increases exponentially with the number of overlap configurations. Alternatively, one could use the factorized state representation of *factorial* HMMs [4]. These models, however, assume a fixed number of loosely connected processes evolving in parallel, which is not a good match to our genomics domain.

Like HMMs, our method, called *CMN-OP* (conditional Markov networks for overlapping patterns), employs element-specific sub-models and probabilistic constraints on neighboring elements qualitatively expressed in a graph. The key difference between CMN-OP and HMMs is the probability distributions they define for an input sequence. While, as mentioned above, an HMM defines a probability distribution over partitions of the sequence, a CMN-OP defines a probability distribution over all possible *joint* arrangements of elements in an input sequence. Figure 2 illustrates this distinction.

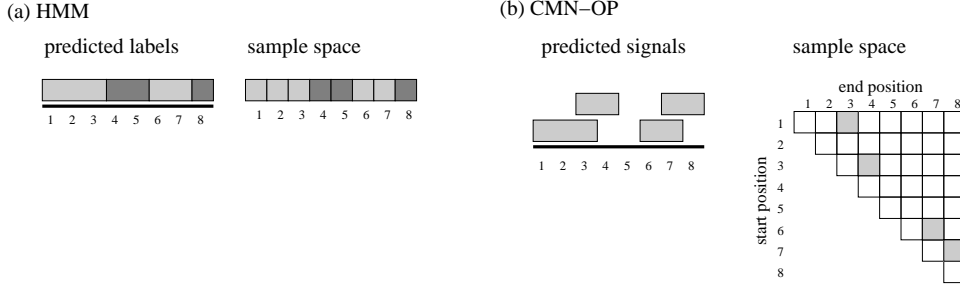

Figure 2: An illustration of the difference in the sample spaces on which probability distributions over labelings are defined by (a) HMMs and (b) CMN-OP models. The left side of (a) shows a sequence of length eight for which an HMM has predicted that an element of interest occupies two subsequences, [1:3] and [6:7]. The darker subsequences, [4:5] and [8:8], represent sequence regions between predicted elements. The right side of (a) shows the corresponding event in the sample space of the HMM, which associates one label with each position. The left side of (b) shows four predicted elements made by a CMN-OP model. The right side of (b) illustrates the corresponding event in the CMN-OP sample space. Each square corresponds to a subsequence, and an event in this sample space assigns a (possibly empty) label to each sub-sequence.

## 2 Models

A *conditional Markov network* [5, 10] (CMN) defines the conditional probability distribution $\Pr(\mathbf{Y}|\mathbf{X})$ where $\mathbf{X}$ is a set of observable input random variables and $\mathbf{Y}$ is a set of output random variables. As with standard Markov networks, a CMN consists of a qualitative graphical component $G = (V, E)$ with vertex set $V$ and edge set $E$ that encodes a set of conditional independence assertions along with a quantitative component in the form of a set of potentials $\Phi$ over the cliques of $G$. In CMNs, $V = \mathbf{X} \cup \mathbf{Y}$. We denote an assignment of values to the set of random variables $\mathbf{U}$ with $\mathbf{u}$. Each clique, $q = (\mathbf{X}_q, \mathbf{Y}_q)$, in the clique set $Q(G)$ has a potential function $\phi_q(\mathbf{x}_q, \mathbf{y}_q) \in \Phi$ that assigns a non-negative number to each of the joint settings of $(\mathbf{X}_q, \mathbf{Y}_q)$. A CMN $(G, \Phi)$ defines the conditional probability distribution $\Pr(\mathbf{y}|\mathbf{x}) = \frac{1}{Z(\mathbf{x})} \prod_{q \in Q(G)} \phi_q(\mathbf{x}_q, \mathbf{y}_q)$ where $Z(\mathbf{x}) = \sum_{\mathbf{y}'} \prod_{q \in \mathcal{Q}(G)} \phi_q(\mathbf{x}_q, \mathbf{y}'_q)$ is the $\mathbf{x}$ dependent normalization factor called the *partition function*. One benefit of CMNs for classification tasks is that they are typically discriminatively trained by maximizing a function based on the conditional likelihood $\Pr(\mathbf{Y}|\mathbf{X})$ over a training set rather than the joint likelihood $\Pr(\mathbf{Y}, \mathbf{X})$.

A common representation for the potentials $\phi_q(\mathbf{y}_q, \mathbf{x}_q)$ is with a log-linear model: $\phi_q(\mathbf{y}_q, \mathbf{x}_q) = \exp\{\sum_b w_q^b f_q^b(\mathbf{y}_q, \mathbf{x}_q)\} = \exp\{\mathbf{w}_q^T \cdot \mathbf{f}_q(\mathbf{y}_q, \mathbf{x}_q)\}$. Here $w_q^b$ is the weight of feature $f_q^b$ and $\mathbf{w}_q$ and $\mathbf{f}_q$ are column vectors of $q$'s weights and features.

Now we show how we use CMNs to predict elements in observation sequences. Given a sequence $\mathbf{x}$ of length $L$, our task is to identify the types and locations of all instances of patterns in $\mathcal{P} = \{P_1, ..., P_N\}$ that are present in $\mathbf{x}$ where $\mathcal{P}$ is a set of pattern types. In the genomics domain $\mathbf{x}$ is a DNA sequence and $\mathcal{P}$ is a set of regulatory elements such as {gene, promoter, terminator}.

A *match m* of a pattern to $\mathbf{x}$ specifies a subsequence $\mathbf{x}_{i:j}$ and a pattern type $P_k \in \mathcal{P}$. We denote the set of all matches of pattern types in $\mathcal{P}$ to $\mathbf{x}$ with $\mathcal{M}(\mathcal{P}, \mathbf{x})$. We call a subset $C = (m_1, m_2, ..., m_M)$ of $\mathcal{M}(\mathcal{P}, \mathbf{x})$ a *configuration*. Matches in $C$ are allowed

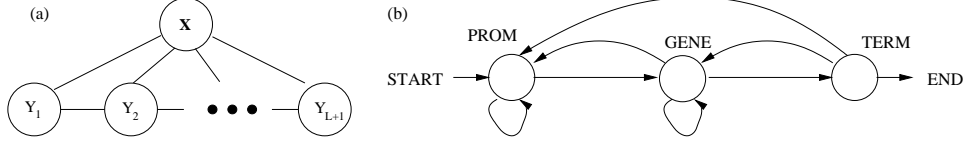

Figure 3: (a) The structure of the CMN-OP induced for the sequence $\mathbf{x}$ of length $L$. The $a^{th}$ pattern match $Y_a$ is conditionally independent of its non-neighbors given its neighbors $\mathbf{X}$, $Y_{a-1}$ and $Y_{a+1}$. (b) The interaction graph we use in the regulatory element prediction task. Vertices are the pattern types along with START and END. Edges connect pattern types that may be adjacent. Edges from START connect to pattern types that may be the first matches Edges into END come from pattern types that may be the last matches.

to overlap however, we assume that no two matches in $C$ have the same start index[1]. Thus, the maximum size of a configuration $C$ is $L$, and the elements of $C$ may be ordered by start position such that $m_a \leq m_{a+1}$. Our models define a conditional probability distribution over configurations given an input sequence $\mathbf{x}$.

Given a sequence $\mathbf{x}$ of length $L$, the output random variables of our models are $\mathbf{Y} = (Y_1, Y_2, ..., Y_L, Y_{L+1})$. We represent a configuration $C = (m_1, m_2, ..., m_M)$ with $\mathbf{Y}$ in the following way. If $a$ is less than or equal to the configuration size $M$, we assign $Y_a$ to the $a^{th}$ match in $C$ ($Y_a = m_a$), otherwise we set $Y_a$ equal to a special value NULL. Note that $Y_{L+1}$ will always be NULL; it is included for notational convenience. Our models define the conditional distribution $\Pr(\mathbf{Y}|\mathbf{X})$.

Our models assume that a pattern match is independent of other matches given its neighbors. That is, $Y_a$ is independent of $Y_{a'}$ for $a' < a - 1$ or $a' > a + 1$ given $\mathbf{X}$, $Y_{a-1}$ and $Y_{a+1}$. This is analogous to the HMM assumption that the next state depends only on the current state. The conditional Markov network structure associated with this assumption is shown in Figure 3(a). The cliques in this graph are $\{Y_a, Y_{a+1}, \mathbf{X}\}$ for $1 \leq a \leq L$. We denote the clique $\{Y_a, Y_{a+1}, \mathbf{X}\}$ with $q_a$.

We define the clique potential of $q_a$ for $a \neq 1$ as the product of a *pattern match* term $g(y_a, \mathbf{x})$ and a *pattern interaction* term $h(y_a, y_{a+1}, \mathbf{x})$. The functions $g()$ and $h()$ are shared among all cliques so $\phi_{q_a}(y_a, y_{a+1}, \mathbf{x}) = g(y_a, \mathbf{x}) \times h(y_a, y_{a+1}, \mathbf{x})$ for $2 \leq a \leq L$. The first clique $q_1$ includes an additional *start placement* term $\alpha(y_1, \mathbf{x})$ that scores the type and position of the first match $y_1$. To ensure that real matches come before any NULL settings and that additional NULL settings do not affect $\Pr(\mathbf{y}|\mathbf{x})$, we require that $g(\text{NULL}, \mathbf{x}) = 1$, $h(\text{NULL},\text{NULL}, \mathbf{x}) = 1$ and $h(\text{NULL}, y_a, \mathbf{x}) = 0$ for all $\mathbf{x}$ and $y_a \neq \text{NULL}$. The pattern match term measures the agreement between the matched subsequence and the pattern type associated with $y_a$. In the genomics domain our representation of the sequence match term is based on regulatory element specific HMMs. The pattern interaction term measures the compatibility between the types and spacing (or overlap) of adjacent matches.

A *Conditional Markov Network for Overlapping Patterns* (CMN-OP) $= (g, h, \alpha)$ specifies a pattern match function $g$, pattern interaction function $h$ and start placement function $\alpha$ that define the conditional distribution $\Pr(\mathbf{y}|\mathbf{x}) = \frac{1}{Z(\mathbf{x})} \prod_{a=1}^{L} \phi_a(q_a, \mathbf{x}) = \frac{\alpha(y_1)}{Z(\mathbf{x})} \prod_{a=1}^{L} g(y_a, \mathbf{x}) h(y_a, y_{a+1}, \mathbf{x})$ where $Z(\mathbf{x})$ is the normalizing partition function. Using the log-linear representation for $g()$ and $h()$ we have $\Pr(\mathbf{y}|\mathbf{x}) = \frac{\alpha(y_1)}{Z(\mathbf{x})} \exp\{\sum_{a=1}^{L} \mathbf{w}_g^T \cdot \mathbf{f}_g(y_a, \mathbf{x}) + \mathbf{w}_h^T \cdot \mathbf{f}_h(y_a, y_{a+1}, \mathbf{x})\}$. Here $\mathbf{w_g}$, $\mathbf{f_g}$, $\mathbf{w_h}$ and $\mathbf{f_h}$ are $g()$ and $h()$'s weights and features.

## 2.1 Representation

Our representation of the pattern match function $g()$ is based on HMMs. We construct an HMM with parameters $\Theta_k$ for each pattern type $P_k$ along with a single background HMM with parameters $\Theta_B$. The pattern match score of $y_a \neq$ NULL with subsequence $\mathbf{x}_{i:j}$ and pattern type $P_k$ is the odds $\Pr(\mathbf{x}_{i:j}|\Theta_k)/\Pr(\mathbf{x}_{i:j}|\Theta_B)$. We have a feature $f_g^k(y_a, \mathbf{x})$ for each pattern type $P_k$ whose value is the logarithm of the odds if the pattern associated with $y_a$ is $P_k$ and zero otherwise. Currently, the weights $\mathbf{w}_g$ are not trained and are fixed at 1. So, $\mathbf{w}_g^T \cdot \mathbf{f}_g(y_a, \mathbf{x}) = f_g^k(y_a, \mathbf{x}) = \log(\Pr(x_{i:j}|\Theta_k)/\Pr(x_{i:j}|\Theta_B))$ where $P_k$ is the pattern of $y_a$.

Our representation of the pattern interaction function $h()$ consists of two components: (i) a directed graph $I$ called the *interaction graph* that contains a vertex for each pattern type in $\mathcal{P}$ along with special vertices START and END and (ii) a set of weighted features for each edge in $I$. The interaction graph encodes qualitative domain knowledge about allowable orderings of pattern types. The value of $h(y_a, y_{a+1}, \mathbf{x}) = \mathbf{w}_h^T \cdot \mathbf{f}_h(y_a, y_{a+1}, \mathbf{x})$ is non-zero only if there is an edge in $I$ from the pattern type associated with $y_a$ to the pattern type associated with $y_{a+1}$. Thus, any configuration with non-zero probability corresponds to a path through $I$. Figure 3(b) shows the interaction graph we use to predict bacterial regulatory elements. It asserts that between the start positions of two genes there may be no element starts, a single terminator start or zero or more promoter starts with the requirement that all promoters start after the start of the terminator. Note that in CMN-OP models, the interaction graph indicates legal orderings over the *start position* of matches not over complete matches as in an HMM.

Each of the pattern interaction features $f \in \mathbf{f}_h$ is associated with an edge in the interaction graph $I$. Each edge $e$ in $I$ has single *bias feature* $f_e^b$ and a set of *distance features* $\mathbf{f}_e^D$. The value of $f_e^b(y_a, y_{a+1}, \mathbf{x})$ is 1 if the pattern types connected by $e$ correspond to the types associated with $y_a$ and $y_{a+1}$ and 0 otherwise. The distance features for edge $e$ provide a discretized representation of the distance between (or amount of overlap of) two adjacent matches of types consistent with $e$. We associate each distance feature $f_e^r \in \mathbf{f}_e^D$ with a range $r$. The value of $f_e^r(y_a, y_{a+1}, \mathbf{x})$ is 1 if the (possibly negative) difference between the start position of $y_{a+1}$ and the end position of $y_a$ is in $r$, otherwise it is 0. The set of ranges for a given edge are non-overlapping. So, $h(y_a, y_{a+1}, \mathbf{x}) = \exp(\mathbf{w}_h^T \cdot \mathbf{f}_h(y_a, y_{a+1}, \mathbf{x})) = \exp(w_e^b + w_e^r)$ where $e$ is the edge for $y_a$ and $y_{a+1}$, $w_e^b$ is the weight of the bias feature $f_e^b$ and $w_e^r$ is the weight of the single distance feature $f_e^r$ whose range contains the spacing between the matches of $y_a$ and $y_{a+1}$.

## 3 Inference and Training

Given a trained model with weights $\mathbf{w}$ and an input sequence $\mathbf{x}$, the inference task is to determine properties of the distribution $\Pr(\mathbf{y}|\mathbf{x})$. Since the cliques of a CMN-OP form a chain we could perform exact inference with the *belief propagation* (BP) algorithm [8]. The number of joint settings in one clique grows $O(L^4)$, however, giving BP a running time of $O(L^5)$ and which is impractical for longer sequences. The exact inference procedure we use, which is inspired the energy minimization algorithm for pictorial structures [3], runs in $O(L^2)$ time.

Our inference procedure exploits two properties of our representation of the pattern interaction function $h()$. First, we use the invariance of $h(y_a, y_{a+1}, x)$ to the start position of $y_a$ and the end position of $y_{a+1}$. In this section, we make this explicit by writing $h(y_a, y_{a+1}, x)$ as $h(k, k', d)$ where $k$ and $k'$ are the pattern types of $y_a$ and

$y_{a+1}$ respectively and $d$ is the distance between (or overlap of if negative) $y_a$ and $y_{a+1}$. The second property we use is the fact that the difference between $h(k, k', d)$ and $h(k, k', d+1)$ is non-zero only if $d$ is the maximum value of the range of one of the distance features $f_e^r \in \mathbf{f}_e^D$ associated with the edge $e = k \to k'$

The inference procedure we use for our CMN-OP models consists of a forward pass and a backward pass. Due to space limitations, we only describe the key aspects of the forward pass. The forward pass fills an $L \times L \times N$ matrix $F$ where we define $F(i, j, k)$ to be the sum of the scores of all partial configurations $\tilde{\mathbf{y}}$ that end with $y^*$ where $y^*$ is the match of $\mathbf{x_{i:j}}$ to $P_k$: $F(i, j, k) \equiv g(y^*, \mathbf{x}) \sum_{\tilde{\mathbf{y}}} \alpha(y_1, \mathbf{x}) \prod_{y_a \in (\tilde{\mathbf{y}} \backslash y^*)} g(y_a, \mathbf{x}) h(y_a, y_{a+1}, \mathbf{x})$ Here $\tilde{\mathbf{y}} = (y_1, y_2, ..., y^*)$ and $\backslash$ denotes set difference.

$F$ has a recursive formulation:

$$F(i, j, k) = g_k(y^*, \mathbf{x}) \left\{ \alpha_k(i) + \sum_{i'=1}^{i-1} \sum_{j'=i'}^{L} \sum_{k'=1}^{N} F(i', j', k') h(k', k, i - j') \right\}.$$

The triple sum is over all possible adjacent previous matches. Due to the first property of $h$ just discussed, the value of the triple sum for setting $F(i, j, k)$ and $F(i, j', k)$ is the same for any $j'$. We cache the value of the triple sum in the $L \times N$ matrix $F_{in}$ where $F_{in}(i, k)$ holds the value needed for setting $F(i, j', k)$ for any $j'$.

We begin the forward pass with $i = 1$ and set the values of $F(1, j, k)$ for all $j$ and $k$ before incrementing $i$. After $i$ is incremented, we use the second property of $h$ to update $F_{in}$ in time $O(N^2 B)$, which is independent of the sequence length $L$, where $B$ is the number of "bins" used in our discretized represenation of distance. The overall time complexity of the forward pass is $O(LN^2 B + L^2 N)$. The first term is for updating $F_{in}$ and the second term is for the constant time setting of the $O(L^2 N)$ elements of $F$. If the sequence length $L$ dominates $N$ and $B$, as it does in the gene regulation domain, the effective running time is $O(L^2)$.

Training involves estimating the weights $\mathbf{w}$ from a training set $D$. An element $d$ of $D$ is a pair $(\mathbf{x}_d, \hat{\mathbf{y}}_d)$ where $\mathbf{x}_d$ is a fully observable sequence and $\hat{\mathbf{y}}_d$ is a partially observable configuration for $\mathbf{x}_d$. To help avoid overfitting we assume a zero-mean Gaussian prior over the weights and optimize the log of the MAP objective function following Taskar et al. [10]: $L(\mathbf{w}, D) = \sum_{d \in D} (\log \Pr(\hat{\mathbf{y}_d} | \mathbf{x}_d)) - \frac{\mathbf{w}^T \cdot \mathbf{w}}{2\sigma^2}$.

The value of the gradient $\nabla L(\mathbf{w}, D)$ in the direction of weight $w \in \mathbf{w}$ is: $\frac{\partial L(\mathbf{w}, D)}{\partial w} = \sum_{d \in D} (E[C_w | \mathbf{x}_d, \hat{\mathbf{y}}_d] - E[C_w | \mathbf{x}_d]) - \frac{w}{\sigma^2}$ where $C_w$ is a random variable representing the number of times the binary feature of $w$ is 1. The expectation is relative to $\Pr(\mathbf{y} | \mathbf{x})$ defined by the current setting of $\mathbf{w}$. The value in the summation is the difference in the expected number of times $w$ is used given both $\mathbf{x}$ and $\hat{\mathbf{y}}$ to the expected number of times $w$ is used given just $\mathbf{x}$. The last term is the shrinking effect of the prior. With the gradient in hand, we can use any of a number of optimization procedures to set $\mathbf{w}$. We use the quasi-Newton method BFGS [6].

## 4 Empirical Evaluation

In this section we evaluate our Markov network approach by applying it to recognize regulatory signals in the *E. coli* genome. Our hypothesis is that the CMN-OP models will provide more accurate predictions than either of two baselines: (i) predicting the signals independently, and (ii) predicting the signals using an HMM.

All three approaches we evaluate – the Markov networks and the two baselines – employ two submodels [1]. The first submodel is an HMM that is used to predict

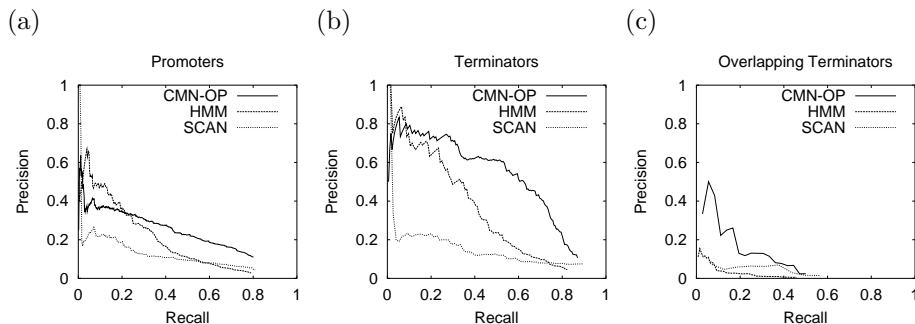

Figure 4: Precision-recall curves for the CMN-OP, HMM and SCAN models on (a) the promoter localization task, (b) the terminator localization task and (c) the terminator localization task for terminators known to overlap genes or promoters.

candidate promoters and the second submodel is a stochastic context free grammar (SCFG) that is used to predict candidate terminators. The first baseline approach, which we refer to as SCAN, involves "scanning" a promoter model and a terminator model along each sequence being processed, and at each position producing a score indicating the likelihood that a promoter or terminator starts at that position. With this baseline, each prediction is made independently of all other predictions. The second baseline is an HMM, similar to the one depicted in Figure 1(b). The HMM that we use here, does not contain the gene submodel shown in Figure 1(b) because the sequences we use in our experiments do not contain entire genes. We have the HMM and CMN-OP models make terminator and promoter predictions for each position in each test sequence. We do this using *posterior decoding* which involves having a model compute the probability that a promoter (terminator) ends at a specified position given that the model somehow explains the sequence.

The data set we use consists of 2,876 subsequences of the *E. coli* genome that collectively contain 471 known promoters and 211 known terminators. Using ten-fold cross-validation, we evaluate the three methods by considering how well each method is able to *localize* predicted promoters and terminators in the test sequences. Under this evaluation criterion, a correct prediction predicts a promoter (terminator) within $k$ bases of an actual promoter (terminator). We set $k$ to 10 for promoters and to 25 for terminators. For all methods, we plot precision-recall (PR) curves by varying a threshold on the prediction confidences. *Recall* is defined as $\frac{TP}{TP+FN}$, and *precision* is defined as $\frac{TP}{TP+FP}$, where $TP$ is the number of true positive predictions, $FN$ is the number of false negatives, and $FP$ is the number of false positives.

Figures 4(a) and 4(b) show PR curves for the promoter and terminator localization tasks, respectively. For both cases, the HMM and CMN-OP models are clearly superior to the SCAN models. This result indicates the value of taking the regularities of relationships among these signals into account when making predictions. For the case of localizing terminators, the CMN-OP PR curve dominates the curve for the HMMs. The difference is not so marked for promoter localization, however. Although the CMN-OP curve is better at high recall levels, the HMM curve is somewhat better at low recall levels. Overall, we conclude that these results show the benefits of representing relationships among predicted signals (as is done in the HMMs and CMN-OP models) and being able to represent and predict overlapping signals. Figure 4(c) shows the PR curves specifically for a set of filtered test sets in which each actual terminator overlaps either a gene or a promoter. These curves indicate that the CMN-OP models have a particular advantage in these cases.

# 5 Conclusion

We have presented an approach, based on Markov networks, able to naturally represent and predict overlapping sequence elements. Our approach first generates a set of candidate elements by having a set of models independently make predictions. Then, we construct a Markov network to decide which candidate predictions to accept. We have empirically validated our approach by using it to recognize promoter and terminator "signals" in a bacterial genome. Our experiments demonstrate that our approach provides more accurate predictions than baseline HMM models.

Although we describe and evaluate our approach in the context of genomics, we believe that it has other applications as well. Consider, for example, the task of segmenting and indexing audio and video streams [7]. We might want to annotate segments of a stream that correspond to specific types of events or to particular individuals who appear or are speaking. Clearly, there might be overlapping events and appearances of people, and moreover, there are likely to be dependencies among events and appearances. Any problem with these two properties is a good candidate for our Markov-network approach.

### Acknowledgments

This research was supported in part by NSF grant IIS-0093016, and NIH grants T15-LM07359-01 and R01-LM07050-01.

## Footnotes

[1]We only need to require configurations to be ordered sets. We make this slightly more stringent assumption to simplify the description of the model.

# References

[1] J. Bockhorst, Y. Qiu, J. Glasner, M. Liu, F. Blattner, and M. Craven. Predicting bacterial transcription units using sequence and expression data. *Bioinformatics*, 19(Suppl. 1):i34–i43, 2003.

[2] M. Ermolaeva, H. Khalak, O. White, H. Smith, and S. Salzberg. Prediction of transcription terminators in bacterial genomes. *J. of Molecular Biology*, 301:27–33, 2000.

[3] P. Felzenszwalb and D. Huttenlocher. Efficient matching of pictorial structures. In *Proc. of the 2000 IEEE Conf. on Computer Vision and Pattern Recognition*, 66–75.

[4] Z. Ghahramani and M. I. Jordan. Factorial hidden markov models. *Machine Learning*, 29:245–273, 1997.

[5] J. Lafferty, A. McCallum, and F. Pereira. Conditional random fields: Probabilistic models for segmenting and labeling sequence data. In *Proc. of the 18th Internat. Conf. on Machine Learning*, pages 282–289, Williamstown, MA, 2001. Morgan Kaufmann.

[6] R. Malouf. A comparison of algorithms for maximum entropy parameter estimation. *Sixth workshop on computational language learning (CoNLL)*, 2002.

[7] National Institute of Standards and Technology. TREC video retrieval evaluation (TRECVID), 2004. http://www-nlpir.nist.gov/projects/t01v/.

[8] J. Pearl. *Probabalistic Reasoning in Intelligent Systems: Networks of Plausible Inference*. Morgan Kaufmann, San Mateo, CA, 1988.

[9] A. Pedersen, P. Baldi, S. Brunak, and Y. Chauvin. Characterization of prokaryotic and eukaryotic promoters using hidden Markov models. In *Proc. of the 4th International Conf. on Intelligent Systems for Molecular Biology*, pages 182–191, St. Louis, MO, 1996. AAAI Press.

[10] B. Taskar, P. Abbeel, and D. Koller. Discriminative probabilistic models for relational data. In *Proc. of the 18th International Conf. on Uncertainty in Artificial Intelligence*, Edmonton, Alberta, 2002. Morgan Kaufmann.

[11] T. Yada, Y. Totoki, T. Takagi, and K. Nakai. A novel bacterial gene-finding system with improved accuracy in locating start codons. *DNA Research*, 8(3):97–106, 2001.
